# A Bayesian Network for Real-Time Musical Accompaniment

**Christopher Raphael**
Department of Mathematics and Statistics,
University of Massachusetts at Amherst,
Amherst, MA 01003-4515,
raphael@math.umass.edu

## Abstract

We describe a computer system that provides a real-time musical accompaniment for a live soloist in a piece of non-improvised music for soloist and accompaniment. A Bayesian network is developed that represents the joint distribution on the times at which the solo and accompaniment notes are played, relating the two parts through a layer of hidden variables. The network is first constructed using the rhythmic information contained in the musical score. The network is then trained to capture the musical interpretations of the soloist and accompanist in an off-line rehearsal phase. During live accompaniment the learned distribution of the network is combined with a real-time analysis of the soloist's acoustic signal, performed with a hidden Markov model, to generate a musically principled accompaniment that respects all available sources of knowledge. A live demonstration will be provided.

## 1   Introduction

We discuss our continuing work in developing a computer system that plays the role of a musical accompanist in a piece of non-improvisatory music for soloist and accompaniment. The system begins with the musical score to a given piece of music. Then, using training for the accompaniment part as well as a series of rehearsals, we learn a performer-specific model for the rhythmic interpretation of the composition. In performance, the system takes the acoustic signal of the live player and generates the accompaniment around this signal, in real-time, while respecting the learned model and the constraints imposed by the score. The accompaniment played by our system responds both flexibly and expressively to the soloist's musical interpretation.

Our system is composed of two high level tasks we call "Listen" and "Play." Listen takes as input the acoustic signal of the soloist and, using a hidden Markov model, performs a real-time analysis of the signal. The output of Listen is essentially a running commentary on the acoustic input which identifies note boundaries in the solo part and communicates these events with variable latency. The HMM framework is well-suited to the listening task and has several attributes we regard

as indispensable to any workable solution:

1. The HMM allows unsupervised training using the Baum-Welch algorithm. Thus we can automatically adapt to changes in solo instrument, microphone placement, ambient noise, room acoustics, and the sound of the accompaniment instrument.

2. Musical accompaniment is inherently a real-time problem. Fast dynamic programming algorithms provide the computational efficiency necessary to process the soloist's acoustic signal at a rate consistent with the real-time demands of our application.

3. Musical signals are occasionally ambiguous locally in time, but become easier to parse when more context is considered. Our system owes much of its accuracy to the probabilistic formulation of the HMM. This formulation allows one to compute the probability that an event is in the past. We delay the estimation of the precise location of an event until we are reasonably confident that it is, in fact, past. In this way our system achieves accuracy while retaining the lowest latency possible in the identification of musical events.

Our work on the Listen component is documented thoroughly in [1] and we omit a more detailed discussion here.

The heart of our system, the Play component, develops a Bayesian network consisting of hundreds of Gaussian random variables including both observable quantities, such as note onset times, and unobservable quantities, such as local tempo. The network can be trained during a rehearsal phase to model both the soloist's and accompanist's interpretations of a specific piece of music. This model then forms the backbone of a principled *real-time* decision-making engine used in performance. We focus here on the Play component which is the most challenging part of our system. A more detailed treatment of various aspects of this work is given in [2–4].

## 2 Knowledge Sources

A musical accompaniment requires the synthesis of a number of different knowledge sources. From a modeling perspective, the fundamental challenge of musical accompaniment is to express these disparate knowledge sources in terms of a common denominator. We describe here the three knowledge sources we use.

1. We work with non-improvisatory music so naturally the musical score, which gives the pitches and relative durations of the various notes, as well as points of synchronization between the soloist and accompaniment, must figure prominently in our model. The score should not be viewed as a rigid grid prescribing the precise times at which musical events will occur; rather, the score gives the basic elastic material which will be stretched in various ways to to produce the actual performance. The score simply does not address most interpretive aspects of performance.

2. Since our accompanist must follow the soloist, the output of the Listen component, which identifies note boundaries in the solo part, constitutes our second knowledge source. While most musical events, such as changes between neighboring diatonic pitches, can be detected very shortly after the change of note, some events, such as rearticulations and octave slurs, are much less obvious and can only be precisely located with the benefit of longer term hindsight. With this in mind, we feel that any successful

accompaniment system cannot synchronize in a purely responsive manner. Rather it must be able to predict the future using the past and base its synchronization on these predictions, as human musicians do.

3. While the same player's performance of a particular piece will vary from rendition to rendition, many aspects of musical interpretation are clearly established with only a few repeated examples. These examples, both of solo performances and human (MIDI) performances of the accompaniment part constitute the third knowledge source for our system. The solo data is used primarily to teach the system how to predict the future evolution of the solo part. The accompaniment data is used to learn the musicality necessary to bring the accompaniment to life.

We have developed a probabilistic model, a Bayesian network, that represents all of these knowledge sources through a jointly Gaussian distribution containing hundreds of random variables. The observable variables in this model are the estimated soloist note onset times produced by Listen and the directly observable times for the accompaniment notes. Between these observable variables lies a layer of hidden variables that describe unobservable quantities such as local tempo, change in tempo, and rhythmic stress.

## 3    A Model for Rhythmic Interpretation

We begin by describing a model for the sequence of note onset times generated by a monophonic (single voice) musical instrument playing a known piece of music. For each of the notes, indexed by $n = 0, \ldots, N$, we define a random vector representing the time, $t_n$, (in seconds) at which the note begins, and the local "tempo," $s_n$, (in secs. per measure) for the note. We model this sequence of random vectors through a random difference equation:

$$\begin{pmatrix} t_{n+1} \\ s_{n+1} \end{pmatrix} = \begin{pmatrix} 1 & l_n \\ 0 & 1 \end{pmatrix} \begin{pmatrix} t_n \\ s_n \end{pmatrix} + \begin{pmatrix} \tau_n \\ \sigma_n \end{pmatrix} \tag{1}$$

$n = 0, \ldots, N - 1$, where $l_n$ is the musical length of the $n^{\text{th}}$ note, in measures, and the $\{(\tau_n, \sigma_n)^t\}$ and $(t_0, s_0)^t$ are mutually independent Gaussian random vectors.

The distributions of the $\{\sigma_n\}$ will tend concentrate around 0 expressing the notion that tempo changes are gradual. The means and variances of the $\{\sigma_n\}$ show where the soloist is speeding-up (negative mean), slowing-down (positive mean), and tell us if these tempo changes are nearly deterministic (low variance), or quite variable (high variance). The $\{\tau_n\}$ variables describe stretches (positive mean) or compressions (negative mean) in the music that occur without any actual change in tempo, as in a *tenuto* or *agogic* accent. The addition of the $\{\tau_n\}$ variables leads to a more musically plausible model, since not all variation in note lengths can be explained through tempo variation. Equally important, however, the $\{\tau_n\}$ variables stabilize the model by not forcing the model to explain, and hence respond to, all note length variation as tempo variation.

Collectively, the distributions of the $(\tau_n, \sigma_n)^t$ vectors characterize the solo player's rhythmic interpretation. Both overall tendencies (means) and the repeatability of these tendencies (covariances) are captured by these distributions.

### 3.1    Joint Model of Solo and Accompaniment

In modeling the situation of musical accompaniment we begin with the our basic rhythm model of Eqn. 1, now applied to the *composite rhythm*. More precisely,

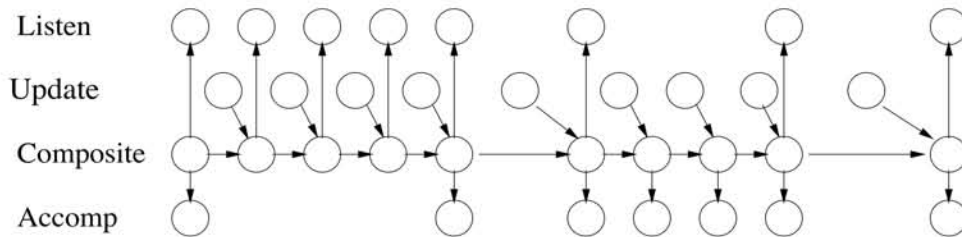

Listen

Update

Composite

Accomp

Figure 1: A graphical description of the dependency structure of our model. The top layer of the graph corresponds to the solo note onset times detected by Listen. The 2nd layer of the graph describes the $(\tau_n, \sigma_n)$ variables that characterize the rhythmic interpretation. The 3rd layer of the graph is the time-tempo process $\{(s_n, t_n)\}$. The bottom layer is the observed accompaniment event times.

let $m_0^s, \ldots, m_{N^s}^s$ and $m_0^a, \ldots, m_{N^a}^a$ denote the positions, in measures, of the various solo and accompaniment events. For example, a sequence of quarter notes in 3/4 time would lie at measure positions 0, 1/3, 2/3, etc. We then let $m_0, \ldots, m_N$ be the sorted union of these two sets of positions with duplicate times removed; thus $m_0 < m_1 < \ldots < m_N$. We then use the model of Eqn. 1 with $l_n = m_{n+1} - m_n$, $n = 0, \ldots, N-1$. A graphical description of this model is given in the middle two layers of Figure 1. In this figure, the layer labeled "Composite" corresponds to the time-tempo variables, $(t_n, s_n)^t$, for the composite rhythm, while the layer labeled "Update" corresponds to the interpretation variables $(\tau_n, \sigma_n)^t$. The directed arrows of this graph indicate the conditional dependency structure of our model. Thus, given all variables "upstream" of a variable, $x$, in the graph, the conditional distribution of $x$ depends only on the parent variables.

Recall that the Listen component estimates the times at which solo notes begin. How do these estimates figure into our model? We model the note onset times estimated by Listen as noisy observations of the true positions $\{t_n\}$. Thus if $m_n$ is a measure position at which a solo note occurs, then the corresponding estimate from Listen is modeled as

$$a_n = t_n + \alpha_n$$

where $\alpha_n \sim N(0, \nu^2)$. Similarly, if $m_n$ is the measure position of an accompaniment event, then we model the observed time at which the event occurs as

$$b_n = t_n + \beta_n$$

where $\beta_n \sim N(0, \eta^2)$. These two collections of observable variables constitute the top layer of our figure, labeled "Listen," and the bottom layer, labeled "Accomp." There are, of course, measure positions at which both solo and accompaniment events should occur. If $n$ indexes such a time then $a_n$ and $b_n$ will both be noisy observations of the true time $t_n$. The vectors/variables $\{(t_0, s_0)^t, (\tau_n, \sigma_n)^t, \alpha_n, \beta_n\}$ are assumed to be mutually independent.

## 4   Training the Model

Our system learns its rhythmic interpretation by estimating the parameters of the $(\tau_n, \sigma_n)$ variables. We begin with a collection of $J$ performances of the accompaniment part played in isolation. We refer to the model learned from this accompaniment data as the "practice room" distribution since it reflects the way the accompanist plays when the constraint of following the soloist is absent. For each

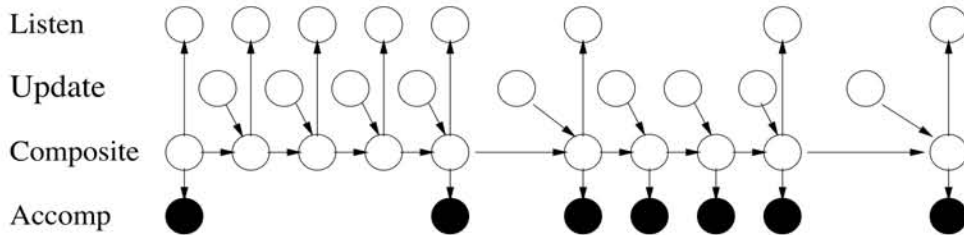

Figure 2: Conditioning on the observed accompaniment performance (darkened circles), we use the message passing algorithm to compute the conditional distributions on the unobservable $\{\tau_n, \sigma_n\}$ variables.

such performance, we treat the sequence of times at which accompaniment events occur as observed variables in our model. These variables are shown with darkened circles in Figure 2. Given an initial assignment of of means and covariances to the $(\tau_n, \sigma_n)$ variables, we use the "message passing" algorithm of Bayesian Networks [8,9] to compute the conditional distributions (given the observed performance) of the $(\tau_n, \sigma_n)$ variables. Several such performances lead to several such estimates, enabling us to improve our initial estimates by reestimating the $(\tau_n, \sigma_n)$ parameters from these conditional distributions.

More specifically, we estimate the $(\tau_n, \sigma_n)$ parameters using the EM algorithm, as follows, as in [7]. We let $\mu_n^0, \Sigma_n^0$ be our initial mean and covariance matrix for the vector $(\tau_n, \sigma_n)$. The conditional distribution of $(\tau_n, \sigma_n)$ given the $j$th accompaniment performance, and using $\{\mu_n^i, \Sigma_n^i\}$, has a $N(m_{j,n}^i, S_n^i)$ distribution where the $m_{j,n}^i$ and $S_n^i$ parameters are computed using the message passing algorithm. We then update our parameter estimates by

$$\mu_n^{i+1} = \frac{1}{J}\sum_{j=1}^{J} m_{j,n}^i$$

$$\Sigma_n^{i+1} = S_n^i + \frac{1}{J}\sum_{j=1}^{J}(m_{j,n}^i - \mu_n^{i+1})(m_{j,n}^i - \mu_n^{i+1})^t$$

The conventional wisdom of musicians is that the accompaniment should follow the soloist. In past versions of our system we have explicitly modeled the asymmetric roles of soloist and accompaniment through a rather complicated graph structure [2–4]. At present we deal with this asymmetry in a more *ad hoc*, however, perhaps more effective, manner, as follows.

Training using the accompaniment performances allows our model to learn some of the musicality these performances demonstrate. Since the soloist's interpretation must take precedence, we want to use this accompaniment interpretation only to the extent that it does not conflict with that of the soloist. We accomplish this by first beginning with the result of the accompaniment training described above. We use the practice room distributions, (the distributions on the $\{(\tau_n, \sigma_n)\}$ learned from the accompaniment data), as the initial distributions, $\{\mu_n^0, \Sigma_n^0\}$. We then run the EM algorithm as described above now treating the currently available collection of *solo* performances as the observed data. During this phase, only those parameters relevant to the soloist's rhythmic interpretation will be modified significantly. Parameters describing the interpretation of a musical segment in which the soloist is mostly absent will be largely unaffected by the second training pass.

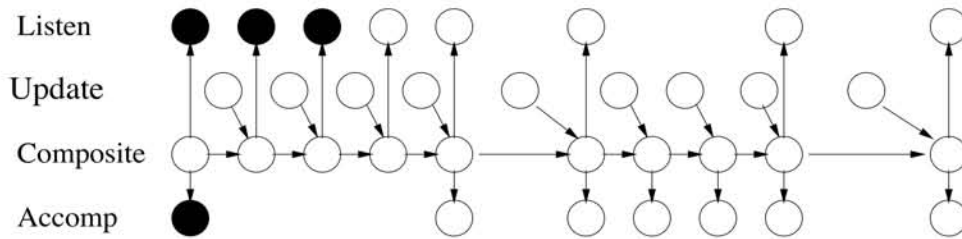

Figure 3: At any given point in the performance we will have observed a collection of solo note times estimated estimated by Listen, and the accompaniment event times (the darkened circles). We compute the conditional distribution on the next unplayed accompaniment event, given these observations.

This solo training actually happens over the course of a series of rehearsals. We first initialize our model to the practice room distribution by training with the accompaniment data. Then we iterate the process of creating a performance with our system, (described in the next section), extracting the sequence of solo note onset times in an off-line estimation process, and then retraining the model using all currently available solo performances. In our experience, only a few such rehearsals are necessary to train a system that responds gracefully and anticipates the soloist's rhythmic nuance where appropriate — generally less than 10.

## 5   Real Time Accompaniment

The methodological key to our real-time accompaniment algorithm is the computation of (conditional) marginal distributions facilitated by the message-passing machinery of Bayesian networks. At any point during the performance some collection of solo notes and accompaniment notes will have been observed, as in Fig. 3. Conditioned on this information we can compute the distribution on the next unplayed accompaniment. The real-time computational requirement is limited by passing only the messages necessary to compute the marginal distribution on the pending accompaniment note.

Once the conditional marginal distribution of the pending accompaniment note is calculated we schedule the note accordingly. Currently we schedule the note to be played at the conditional mean time, given all observed information, however other reasonable choices are possible. Note that this conditional distribution depends on all of the sources of information included in our model: The score information, all currently observed solo and accompaniment note times, and the rhythmic interpretations demonstrated by both the soloist and accompanist captured during the training phase.

The initial scheduling of each accompaniment note takes place immediately after the previous accompaniment note is played. It is possible that a solo note will be detected before the pending accompaniment is played; in this event the pending accompaniment event is rescheduled by recomputing the its conditional distribution using the newly available information. The pending accompaniment note is rescheduled each time an additional solo note is detected until its currently scheduled time arrives, at which time it is finally played. In this way our accompaniment makes use of all currently available information.

Does our system pass the musical equivalent of the Turing Test? We presume no more objectivity in answering this question than we would have in judging

the merits of our other children. However, we believe that the level of musicality attained by our system is truly surprising, while the reliability is sufficient for live demonstration. We hope that the interested reader will form an independent opinion, even if different from ours, and to this end we have made musical examples demonstrating our progress available on the web page: http://fafner.math.umass.edu/music_plus_one.

## Acknowledgments

This work supported by NSF grants IIS-998789 and IIS-0113496.

## References

[1] Raphael C. (1999), "Automatic Segmentation of Acoustic Musical Signals Using Hidden Markov Models," *IEEE Transactions on Pattern Analysis and Machine Intelligence*, Vol. 21, No. 4, pp. 360–370.

[2] Raphael C. (2001), "A Probabilistic Expert System for Automatic Musical Accompaniment," *Journal of Computational and Graphical Statistics*, vol. 10 no. 3, 487–512.

[3] Raphael C. (2001), "Can the Computer Learn to Play Expressively?" *Proceedings of Eighth International Workshop on Artificial Intelligence and Statistics*, 113–120, Morgan Kauffman.

[4] Raphael C. (2001), "Synthesizing Musical Accompaniments with Bayesian Belief Networks," *Journal of New Music Research*, vol. 30, no. 1, 59–67.

[5] Spiegelhalter D., Dawid A. P., Lauritzen S., Cowell R. (1993), "Bayesian Analysis in Expert Systems," *Statistical Science*, Vol. 8, No. 3, pp. 219–283.

[6] Cowell R., Dawid A. P., Lauritzen S., Spiegelhalter D. (1999), "Probabilistic Networks and Expert Systems," Springer, New York.

[7] Lauritzen S. L. (1995), "The EM Algorithm for Graphical Association Models with Missing Data," *Computational Statistics and Data Analysis*, Vol. 19, pp. 191–201.

[8] Lauritzen S. L. (1992), "Propagation of Probabilities, Means, and Variances in Mixed Graphical Association Models," *Journal of the American Statistical Association*, Vol. 87, No. 420, (Theory and Methods), pp. 1098–1108.

[9] Lauritzen S. L. and F. Jensen (1999), "Stable Local Computation with Conditional Gaussian Distributions," *Technical Report R-99-2014*, Department of Mathematic Sciences, Aalborg University.